# Learning optimal spike-based representations

**Ralph Bourdoukan**\*
Group for Neural Theory
École Normale Supérieure
Paris, France
ralph.bourdoukan@ens.fr

**David G.T. Barrett**\*
Group for Neural Theory
École Normale Supérieure
Paris, France
david.barrett@ens.fr

**Christian K. Machens**
Champalimaud Neuroscience Programme
Champalimaud Centre for the Unknown
Lisbon, Portugal
christian.machens@neuro.fchampalimaud.org

**Sophie Denève**
Group for Neural Theory
École Normale Supérieure
Paris, France
sophie.deneve@ens.fr

## Abstract

How can neural networks learn to represent information optimally? We answer this question by deriving spiking dynamics and learning dynamics directly from a measure of network performance. We find that a network of integrate-and-fire neurons undergoing Hebbian plasticity can learn an optimal spike-based representation for a linear decoder. The learning rule acts to minimise the membrane potential magnitude, which can be interpreted as a representation error after learning. In this way, learning reduces the representation error and drives the network into a robust, balanced regime. The network becomes balanced because small representation errors correspond to small membrane potentials, which in turn results from a balance of excitation and inhibition. The representation is robust because neurons become self-correcting, only spiking if the representation error exceeds a threshold. Altogether, these results suggest that several observed features of cortical dynamics, such as excitatory-inhibitory balance, integrate-and-fire dynamics and Hebbian plasticity, are signatures of a robust, optimal spike-based code.

A central question in neuroscience is to understand how populations of neurons represent information and how they learn to do so. Usually, learning and information representation are treated as two different functions. From the outset, this separation seems like a good idea, as it reduces the problem into two smaller, more manageable chunks. Our approach, however, is to study these together. This allows us to treat learning and information representation as two sides of a single mechanism, operating at two different timescales.

Experimental work has given us several clues about the regime in which real networks operate in the brain. Some of the most prominent observations are: (a) high trial-to-trial variability—a neuron responds differently to repeated, identical inputs [1, 2]; (b) asynchronous firing at the network level—spike trains of different neurons are at most very weakly correlated [3, 4, 5]; (c) tight balance of excitation and inhibition—every excitatory input is met by an inhibitory input of equal or greater size [6, 7, 8] and (4) spike-timing-dependent plasticity (STDP)—the strength of synapses change as a function of presynaptic and postsynaptic spike times [9].

Previously, it has been shown that observations (a)–(c) can be understood as signatures of an optimal, spike-based code [10, 11]. The essential idea is to derive spiking dynamics from the assumption that neurons only fire if their spike improves information representation. Information in a network may

---

originate from several possible sources: external sensory input, external neural network input, or alternatively, it may originate within the network itself as a memory, or as a computation. Whatever the source, this initial assumption leads directly to the conclusion that a network of integrate-and-fire neurons can optimally represent a signal while exhibiting properties (a)–(c).

A major problem with this framework is that network connectivity must be completely specified *a priori*, and requires the tuning of $N^2$ parameters, where $N$ is the number of neurons in the network. Although this is feasible mathematically, it is unclear how a real network could tune itself into this optimal regime. In this work, we solve this problem using a simple synaptic learning rule. The key insight is that the plasticity rule can be derived from the same basic principle as the spiking rule in the earlier work—namely, that any change should improve information representation.

Surprisingly, this can be achieved with a local, Hebbian learning rule, where synaptic plasticity is proportional to the product of presynaptic firing rates with post-synaptic membrane potentials. Spiking and synaptic plasticity then work hand in hand towards the same goal: the spiking of a neuron decreases the representation error on a fast time scale, thereby giving rise to the actual population representation; synaptic plasticity decreases the representation error on a slower time scale, thereby improving or maintaining the population representation. For a large set of initial connectivities and spiking dynamics, neural networks are driven into a balanced regime, where excitation and inhibition cancel each other and where spike trains are asynchronous and irregular. Furthermore, the learning rule that we derive reproduces the main features of STDP (property (d) above). In this way, a network can learn to represent information optimally, with synaptic, neural and network dynamics consistent with those observed experimentally.

# 1  Derivation of the learning rule for a single neuron

We begin by deriving a learning rule for a single neuron with an autapse (a self-connection) (Fig. 1A). Our approach is to derive synaptic dynamics for the autapse and spiking dynamics for the neuron such that the neuron learns to optimally represent a time-varying input signal. We will derive a learning rule for networks of neurons later, after we have developed the fundamental concepts for the single neuron case.

Our first step is to derive optimal spiking dynamics for the neuron, so that we have a target for our learning rule. We do this by making two simple assumptions [11]. First, we assume that the neuron can provide an estimate or read-out $\hat{x}(t)$ of a time-dependent signal $x(t)$ by filtering its spike train $o(t)$ as follows:

$$\dot{\hat{x}}(t) = -\hat{x}(t) + \Gamma o(t),\tag{1}$$

where $\Gamma$ is a fixed read-out weight, which we will refer to as the neuron's "output kernel" and the spike train can be written as $o(t) = \sum_i \delta(t - t_i)$, where $\{t_i\}$ are the spike times. Next, we assume that the neuron only produces a spike if that spike improves the read-out, where we measure the read-out performance through a simple squared-error loss function:

$$L(t) = \big(x(t) - \hat{x}(t)\big)^2.\tag{2}$$

With these two assumptions, we can now derive optimal spiking dynamics. First, we observe that if the neuron produces an additional spike at time $t$, the read-out increases by $\Gamma$, and the loss function becomes $L(t|\text{spike}) = (x(t) - (\hat{x}(t) + \Gamma))^2$. This allows us to restate our spiking rule as follows: the neuron should only produce a spike if $L(t|\text{no spike}) > L(t|\text{spike})$, or $(x(t) - \hat{x}(t))^2 > (x(t) - (\hat{x}(t) + \Gamma))^2$. Now, squaring both sides of this inequality, defining $V(t) \equiv \Gamma(x(t) - \hat{x}(t))$ and defining $T \equiv \Gamma^2/2$ we find that the neuron should only spike if:

$$V(t) > T.\tag{3}$$

We interpret $V(t)$ to be the membrane potential of the neuron, and we interpret $T$ as the spike threshold. This interpretation allows us to understand the membrane potential functionally: the voltage is proportional to a prediction error—the difference between the read-out $\hat{x}(t)$ and the actual signal $x(t)$. A spike is an error reduction mechanism—the neuron only spikes if the error exceeds the spike threshold. This is a greedy minimisation, in that the neuron fires a spike whenever that action decreases $L(t)$ without considering the future impact of that spike. Importantly, the neuron does not require direct access to the loss function $L(t)$.

To determine the membrane potential dynamics, we take the derivative of the voltage, which gives us $\dot{V} = \Gamma(\dot{x} - \dot{\hat{x}})$. (Here, and in the following, we will drop the time index for notational brevity.) Now, using Eqn. (1) we obtain $\dot{V} = \Gamma\dot{x} - \Gamma(-\hat{x} + \Gamma o) = -\Gamma(x - \hat{x}) + \Gamma(\dot{x} + x) - \Gamma^2 o$, so that:

$$\dot{V} = -V + \Gamma c - \Gamma^2 o, \tag{4}$$

where $c = \dot{x} + x$ is the neural input. This corresponds exactly to the dynamics of a leaky integrate-and-fire neuron with an inhibitory autapse[1] of strength $\Gamma^2$, and a feedforward connection strength $\Gamma$.

The dynamics and connectivity guarantee that a neuron spikes at just the right times to optimise the loss function (Fig. 1B). In addition, it is especially robust to noise of different forms, because of its error-correcting nature. If $x$ is constant in time, the voltage will rise up to the threshold $T$ at which point a spike is fired, adding a delta function to the spike train $o$ at time $t$, thereby producing a read-out $\hat{x}$ that is closer to $x$ and causing an instantaneous drop in the voltage through the autapse, by an amount $\Gamma^2 = 2T$, effectively resetting the voltage to $V = -T$.

We now have a target for learning—we know the connection strength that a neuron must have at the end of learning if it is to represent information optimally, for a linear read-out. We can use this target to derive synaptic dynamics that can learn an optimal representation from experience. Specifically, we consider an integrate-and-fire neuron with some arbitrary autapse strength $\omega$. The dynamics of this neuron are given by

$$\dot{V} = -V + \Gamma c - \omega o. \tag{5}$$

This neuron will not produce the correct spike train for representing $x$ through a linear read-out (Eqn. (1)) unless $\omega = \Gamma^2$.

Our goal is to derive a dynamical equation for the synapse $\omega$ so that the spike train becomes optimal. We do this by quantifying the loss that we are incurring by using the suboptimal strength, and then deriving a learning rule that minimises this loss with respect to $\omega$. The loss function underlying the spiking dynamics determined by Eqn. (5) can be found by reversing the previous membrane potential analysis. First, we integrate the differential equation for $V$, assuming that $\omega$ changes on time scales much slower than the membrane potential. We obtain the following (formal) solution:

$$V = \Gamma x - \omega \bar{o}, \tag{6}$$

where $\bar{o}$ is determined by $\dot{\bar{o}} = -\bar{o} + o$. The solution to this latter equation is $\bar{o} = h * o$, a convolution of the spike train with the exponential kernel $h(\tau) = \theta(\tau)\exp(-\tau)$. As such, it is analogous to the instantaneous firing rate of the neuron.

Now, using Eqn. (6), and rewriting the read-out as $\hat{x} = \Gamma\bar{o}$, we obtain the loss incurred by the sub-optimal neuron,

$$L = (x - \hat{x})^2 = \frac{1}{\Gamma^2}\left(V^2 + 2(\omega - \Gamma^2)\bar{o} + (\omega - \Gamma^2)^2\bar{o}^2\right). \tag{7}$$

We observe that the last two terms of Eqn. (7) will vanish whenever $\omega = \Gamma^2$, i.e., when the optimal reset has been found. We can therefore simplify the problem by defining an alternative loss function,

$$L_V = \frac{1}{2}V^2, \tag{8}$$

which has the same minimum as the original loss ($V = 0$ or $x = \hat{x}$, compare Eqn. (2)), but yields a simpler learning algorithm. We can now calculate how changes to $\omega$ affect $L_V$:

$$\frac{\partial L_V}{\partial \omega} = V\frac{\partial V}{\partial \omega} = -V\bar{o} - V\omega\frac{\partial \bar{o}}{\partial \omega}. \tag{9}$$

We can ignore the last term in this equation (as we will show below). Finally, using simple gradient descent, we obtain a simple Hebbian-like synaptic plasticity rule:

$$\tau\dot{\omega} = -\frac{\partial L_V}{\partial \omega} = V\bar{o}, \tag{10}$$

where $\tau$ is the learning time constant.

This synaptic learning rule is capable of learning the synaptic weight $\omega$ that minimises the difference between $x$ and $\hat{x}$ (Fig. 1B). During learning, the synaptic weight changes in proportion to the post-synaptic voltage $V$ and the pre-synaptic firing rate $\bar{o}$ (Fig. 1C). As such, this is a Hebbian learning rule. Of course, in this single neuron case, the pre-synaptic neuron and post-synaptic neuron are the same neuron. The synaptic weight gradually approaches its optimal value $\Gamma^2$. However, it never completely stabilises, because learning never stops as long as neurons are spiking. Instead, the synapse oscillates closely about the optimal value (Fig. 1D).

This is also a "greedy" learning rule, similar to the spiking rule, in that it seeks to minimise the error at each instant in time, without regard for the future impact of those changes. To demonstrate that the second term in Eqn. (5) can be neglected we note that the equations for $V$, $\bar{o}$, and $\omega$ define a system of coupled differential equations that can be solved analytically by integrating between spikes. This results in a simple recurrence relation for changes in $\omega$ from the $i^{th}$ to the $(i+1)^{th}$ spike,

$$\omega_{i+1} = \omega_i + \frac{\omega_i(\omega_i - 2T)}{\tau(T - \Gamma c - \omega_i)}. \tag{11}$$

This iterative equation has a single stable fixed point at $\omega = 2T = \Gamma^2$, proving that the neuron's autaptic weight or reset will approach the optimal solution.

## 2 Learning in a homogeneous network

We now generalise our learning rule derivation to a network of $N$ identical, homogeneously connected neurons. This generalisation is reasonably straightforward because many characteristics of the single neuron case are shared by a network of identical neurons. We will return to the more general case of heterogeneously connected neurons in the next section.

We begin by deriving optimal spiking dynamics, as in the single neuron case. This provides a target for learning, which we can then use to derive synaptic dynamics. As before, we want our network to produce spikes that optimally represent a variable $x$ for a linear read-out. We assume that the read-out $\hat{x}$ is provided by summing and filtering the spike trains of all the neurons in the network:

$$\dot{\hat{x}} = -\hat{x} + \mathbf{\Gamma o}, \tag{12}$$

where the row vector $\mathbf{\Gamma} = (\Gamma, \ldots, \Gamma)$ contains the read-out weights[2] of the neurons and the column vector $\mathbf{o} = (o_1, \ldots, o_N)$ their spike trains. Here, we have used identical read-out weights for each neuron, because this indirectly leads to homogeneous connectivity, as we will demonstrate.

Next, we assume that a neuron only spikes if that spike reduces a loss-function. This spiking rule is similar to the single neuron spiking rule except that this time there is some ambiguity about which neuron should spike to represent a signal. Indeed, there are many different spike patterns that provide exactly the same estimate $\hat{x}$. For example, one neuron could fire regularly at a high rate (exactly like our previous single neuron example) while all others are silent. To avoid this firing rate ambiguity, we use a modified loss function, that selects amongst all equivalent solutions, those with the smallest neural firing rates. We do this by adding a 'metabolic cost' term to our loss function, so that high firing rates are penalised:

$$L = (x - \hat{x})^2 + \mu\|\bar{\mathbf{o}}\|^2, \tag{13}$$

where $\mu$ is a small positive constant that controls the cost-accuracy trade-off, akin to a regularisation parameter.

Each neuron in the optimal network will seek to reduce this loss function by firing a spike. Specifically, the $i^{th}$ neuron will spike whenever $L(\text{no spike in } i) > L(\text{spike in } i)$. This leads to the following spiking rule for the $i^{th}$ neuron:

$$V_i > T_i \tag{14}$$

where $V_i \equiv \Gamma(x - \hat{x}) - \mu\bar{o}_i$ and $T_i \equiv \Gamma^2/2 + \mu/2$. We can naturally interpret $V_i$ as the membrane potential of the $i^{th}$ neuron and $T_i$ as the spiking threshold of that neuron. As before, we can now derive membrane potential dynamics:

$$\dot{\mathbf{V}} = -\mathbf{V} + \mathbf{\Gamma}^T c - (\mathbf{\Gamma}^T\mathbf{\Gamma} + \mu\boldsymbol{I})\mathbf{o}, \tag{15}$$

where $\boldsymbol{I}$ is the identity matrix and $\boldsymbol{\Gamma}^T \boldsymbol{\Gamma} + \mu \boldsymbol{I}$ is the network connectivity. We can interpret the self-connection terms $\{\Gamma^2 + \mu\}$ as voltage resets that decrease the voltage of any neuron that spikes. This optimal network is equivalent to a network of identical integrate-and-fire neurons with homogeneous inhibitory connectivity.

The network has some interesting dynamical properties. The voltages of all the neurons are largely synchronous, all increasing to the spiking threshold at about the same time[3] (Fig. 1F). Nonetheless, neural spiking is asynchronous. The first neuron to spike will reset itself by $\Gamma^2 + \mu$, and it will inhibit all the other neurons in the network by $\Gamma^2$. This mechanism prevents neurons from spik-

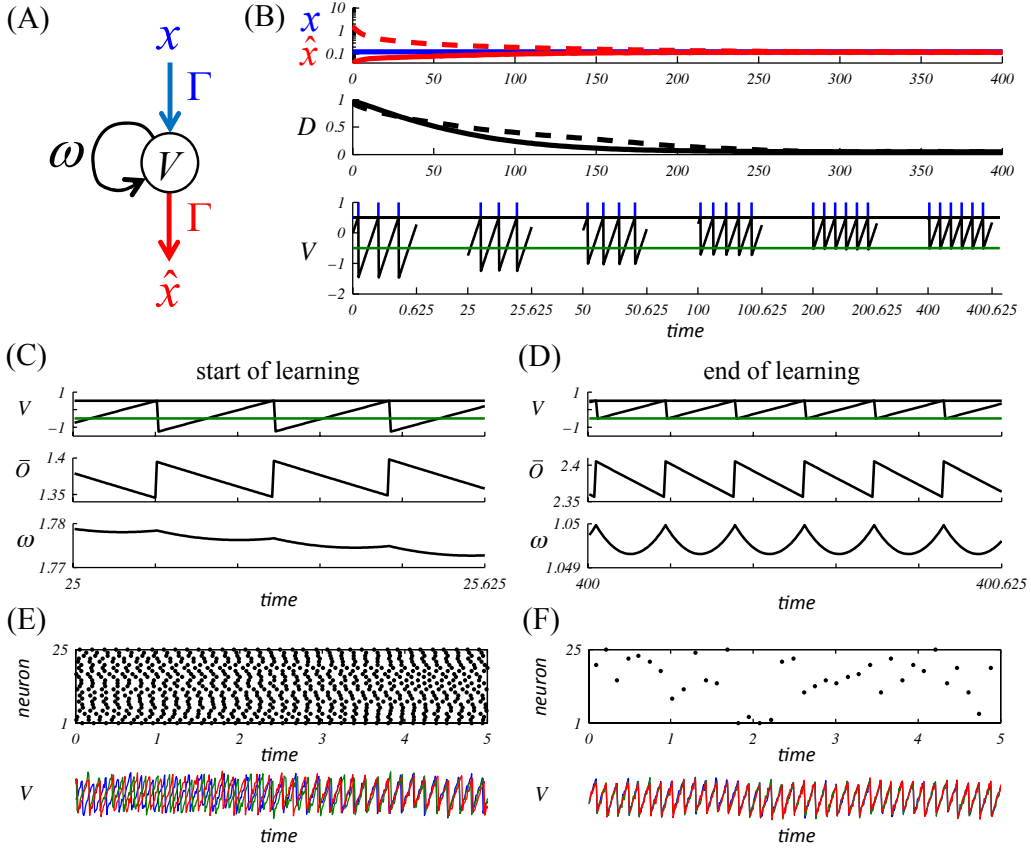

Figure 1: Learning in a single neuron and a homogeneous network. (A) A single neuron represents an input signal $x$ by producing an output $\hat{x}$. (B) During learning, the single neuron output $\hat{x}$ (solid red line, top panel) converges towards the input $x$ (blue). Similarly, for a homogeneous network the output $\hat{x}$ (dashed red line, top panel) converges towards $x$. Connectivity also converges towards optimal connectivity in both the single neuron case (solid black line, middle panel) and the homogeneous network case (dashed black line, middle panel), as quantified by $D = \max_{i,j}(\left|\Omega_{ij} - \Omega_{ij}^{opt}\right|^2 / \left|\Omega_{ij}^{opt}\right|^2)$ at each point in time. Consequently, the membrane potential reset (bottom panel) converges towards the optimal reset (green line, bottom panel). Spikes are indicated by blue vertical marks, and are produced when the membrane potential reaches threshold (bottom panel). Here, we have rescaled time, as indicated, for clarity. (C) Our learning rule dictates that the autapse $\omega$ in our single neuron (bottom panel) changes in proportion to the membrane potential (top panel) and the firing rate (middle panel). (D) At the end of learning, the reset $\omega$ fluctuates weakly about the optimal value. (E) For a homogeneous network, neurons spike regularly at the start of learning, as shown in this raster plot. Membrane potentials of different neurons are weakly correlated. (F) At the end of learning, spiking is very irregular and membrane potentials become more synchronous.

ing synchronously. The population as a whole acts similarly to the single neuron in our previous example. Each neuron fires regularly, even if a different neuron fires in every integration cycle.

The design of this optimal network requires the tuning of $N(N-1)$ synaptic parameters. How can an arbitrary network of integrate-and-fire neurons learn this optimum? As before, we address this question by using the optimal network as a target for learning. We start with an arbitrarily connected network of integrate-and-fire neurons:

$$\dot{\mathbf{V}} = -\mathbf{V} + \mathbf{\Gamma}^T c - \mathbf{\Omega o}, \tag{16}$$

where $\mathbf{\Omega}$ is a matrix of connectivity weights, which includes the resets of the individual neurons. Assuming that learning occurs on a slow time scale, we can rewrite this equation as

$$\mathbf{V} = \mathbf{\Gamma}^T x - \mathbf{\Omega}\bar{\mathbf{o}}. \tag{17}$$

Now, repeating the arguments from the single neuron derivation, we modify the loss function to obtain an online learning rule. Specifically, we set $L_V = \|\mathbf{V}\|^2/2$, and calculate the gradient:

$$\frac{\partial L_V}{\partial \Omega_{ij}} = \sum_k V_k \frac{\partial V_k}{\partial \Omega_{ij}} = -\sum_k V_k \delta_{ki} \bar{o}_j - \sum_{kl} V_k \Omega_{kl} \frac{\partial \bar{o}_l}{\partial \Omega_{ij}}. \tag{18}$$

We can simplify this equation considerably by observing that the contribution of the second summation is largely averaged out under a wide variety of realistic conditions[4]. Therefore, it can be neglected, and we obtain the following local learning rule:

$$\tau \dot{\Omega}_{ij} = -\frac{\partial L_V}{\partial \Omega_{ij}} = V_i \bar{o}_j. \tag{19}$$

This is a Hebbian plasticity rule, whereby connectivity changes in proportion to the presynaptic firing rate $\bar{o}_j$ and post-synaptic membrane potential $V_i$. We assume that the neural thresholds are set to a constant $T$ and that the neural resets are set to their optimal values $-T$. In the previous section we demonstrated that these resets can be obtained by a Hebbian plasticity rule (Eqn. (10)).

This learning rule minimises the difference between the read-out and the signal, by approaching the optimal recurrent connection strengths for the network (Fig. 1B). As in the single neuron case, learning does not stop, so the connection strengths fluctuate close to their optimal value. During learning, network activity becomes progressively more asynchronous as it progresses towards optimal connectivity (Fig. 1E, F).

## 3 Learning in the general case

Now that we have developed the fundamental concepts underlying our learning rule, we can derive a learning rule for the more general case of a network of $N$ arbitrarily connected leaky integrate-and-fire neurons. Our goal is to understand how such networks can learn to optimally represent a $J$-dimensional signal $\mathbf{x} = (x_1, \ldots, x_J)$, using the read-out equation $\dot{\mathbf{x}} = -\mathbf{x} + \mathbf{\Gamma o}$.

We consider a network with the following membrane potential dynamics:

$$\dot{\mathbf{V}} = -\mathbf{V} + \mathbf{\Gamma}^T \mathbf{c} - \mathbf{\Omega o}, \tag{20}$$

where $\mathbf{c}$ is a $J$-dimensional input. We assume that this input is related to the signal according to $\mathbf{c} = \dot{\mathbf{x}} + \mathbf{x}$. This assumption can be relaxed by treating the input as the control for an arbitrary linear dynamical system, in which case the signal represented by the network is the output of such a computation [11]. However, this further generalisation is beyond the scope of this work.

As before, we need to identify the optimal recurrent connectivity so that we have a target for learning. Most generally, the optimal recurrent connectivity is $\mathbf{\Omega}^{\text{opt}} \equiv \mathbf{\Gamma}^T\mathbf{\Gamma} + \mu \mathbf{I}$. The output kernels of the individual neurons, $\mathbf{\Gamma}_i$, are given by the rows of $\mathbf{\Gamma}$, and their spiking thresholds by $T_i \equiv \|\mathbf{\Gamma}_i\|^2/2 +$

$\mu/2$. With these connections and thresholds, we find that a network of integrate-and-fire neurons will produce spike trains in such a way that the loss function $L = \|\mathbf{x} - \hat{\mathbf{x}}\|^2 + \mu\|\bar{\mathbf{o}}\|^2$ is minimised, where the read-out is given by $\hat{\mathbf{x}} = \boldsymbol{\Gamma}\bar{\mathbf{o}}$. We can show this by prescribing a greedy[5] spike rule: a spike is fired by neuron $i$ whenever $L(\text{no spike in } i) > L(\text{spike in } i)$ [11]. The resulting spike generation rule is

$$V_i > T_i, \tag{21}$$

where $V_i \equiv \boldsymbol{\Gamma}_i^T(\mathbf{x} - \hat{\mathbf{x}}) - \mu\bar{o}_i$ is interpreted as the membrane potential.

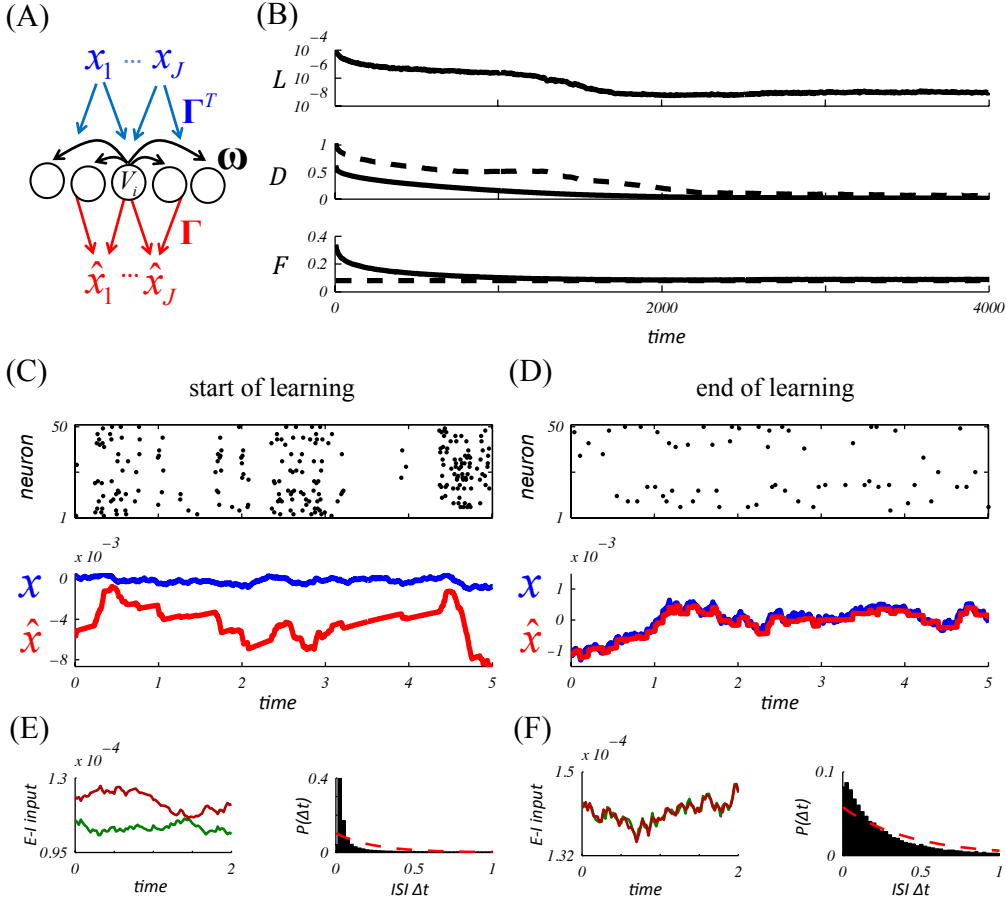

Figure 2: Learning in a heterogeneous network. (A) A network of neurons represents an input signal $\mathbf{x}$ by producing an output $\hat{\mathbf{x}}$. (B) During learning, the loss $L$ decreases (top panel). The difference between the connection strengths and the optimal strengths also decreases (middle panel), as quantified by the mean difference (solid line), given by $D = \left\|\boldsymbol{\Omega} - \boldsymbol{\Omega}^{\text{opt}}\right\|^2 / \left\|\boldsymbol{\Omega}^{\text{opt}}\right\|^2$ and the maximum difference (dashed line), given by $\max_{i,j}\left(\left|\Omega_{ij} - \Omega_{ij}^{\text{opt}}\right|^2 / \left|\Omega_{ij}^{\text{opt}}\right|^2\right)$. The mean population firing rate (solid line, bottom panel) also converges towards the optimal firing rate (dashed line, bottom panel). (C, E) Before learning, a raster plot of population spiking shows that neurons produce bursts of spikes (upper panel). The network output $\hat{\mathbf{x}}$ (red line, middle panel) fails to represent $\mathbf{x}$ (blue line, middle panel). The excitatory input (red, bottom left panel) and inhibitory input (green, bottom left panel) to a randomly selected neuron is not tightly balanced. Furthermore, a histogram of interspike intervals shows that spiking activity is not Poisson, as indicated by the red line that represents a best-fit exponential distribution. (D, F) At the end of learning, spiking activity is irregular and Poisson-like, excitatory and inhibitory input is tightly balanced and $\hat{\mathbf{x}}$ matches $\mathbf{x}$.

How can we learn this optimal connection matrix? As before, we can derive a learning rule by minimising the cost function $L_V = \|\mathbf{V}\|^2/2$. This leads to a Hebbian learning rule with the same form as before:

$$\tau\dot{\Omega}_{ij} = V_i\bar{o}_j. \qquad (22)$$

Again, we assume that the neural resets are given by $-T_i$. Furthermore, in order for this learning rule to work, we must assume that the network input explores all possible directions in the $J$-dimensional input space (since the kernels $\mathbf{\Gamma}_i$ can point in any of these directions). The learning performance does not critically depend on how the input variable space is sampled as long as the exploration is extensive. In our simulations, we randomly sample the input $\mathbf{c}$ from a Gaussian white noise distribution at every time step for the entire duration of the learning.

We find that this learning rule decreases the loss function $L$, thereby approaching optimal network connectivity and producing optimal firing rates for our linear decoder (Fig. 2B). In this example, we have chosen connectivity that is initially much too weak at the start of learning. Consequently, the initial network behaviour is similar to a collection of unconnected single neurons that ignore each other. Spike trains are not Poisson-like, firing rates are excessively large, excitatory and inhibitory input is unbalanced and the decoded variable $\hat{\mathbf{x}}$ is highly unreliable (Fig. 2C, E). As a result of learning, the network becomes tightly balanced and the spike trains become asynchronous, irregular and Poisson-like with much lower rates (Fig. 2D, F). However, despite this apparent variability, the population representation is extremely precise, only limited by the the metabolic cost and the discrete nature of a spike. This learnt representation is far more precise than a rate code with independent Poisson spike trains [11]. In particular, shuffling the spike trains in response to identical inputs drastically degrades this precision.

## 4   Conclusions and Discussion

In population coding, large trial-to-trial spike train variability is usually interpreted as noise [2]. We show here that a deterministic network of leaky integrate-and-fire neurons with a simple Hebbian plasticity rule can self-organise into a regime where information is represented far more precisely than in noisy rate codes, while appearing to have noisy Poisson-like spiking dynamics.

Our learning rule (Eqn. (22)) has the basic properties of STDP. Specifically, a presynaptic spike occurring immediately before a post-synaptic spike will potentiate a synapse, because membrane potentials are positive immediately before a postsynaptic spike. Furthermore, a presynaptic spike occurring immediately after a post-synaptic spike will depress a synapse, because membrane potentials are always negative immediately after a postsynaptic spike. This is similar in spirit to the STDP rule proposed in [12], but different to classical STDP, which depends on post-synaptic spike times [9].

This learning rule can also be understood as a mechanism for generating a tight balance between excitatory and inhibitory input. We can see this by observing that membrane potentials after learning can be interpreted as representation errors (projected onto the read-out kernels). Therefore, learning acts to minimise the magnitude of membrane potentials. Excitatory and inhibitory input must be balanced if membrane potentials are small, so we can equate balance with optimal information representation.

Previous work has shown that the balanced regime produces (quasi-)chaotic network dynamics, thereby accounting for much observed cortical spike train variability [13, 14, 4]. Moreover, the STDP rule has been known to produce a balanced regime [16, 17]. Additionally, recent theoretical studies have suggested that the balanced regime plays an integral role in network computation [15, 13]. In this work, we have connected these mechanisms and functions, to conclude that learning this balance is equivalent to the development of an optimal spike-based population code, and that this learning can be achieved using a simple Hebbian learning rule.

**Acknowledgements**

We are grateful for generous funding from the Emmy-Noether grant of the Deutsche Forschungs-gemeinschaft (CKM) and the Chaire d'excellence of the Agence National de la Recherche (CKM, DB), as well as a James Mcdonnell Foundation Award (SD) and EU grants BACS FP6-IST-027140, BIND MECT-CT-20095-024831, and ERC FP7-PREDSPIKE (SD).

## Footnotes

[1]This contribution of the autapse can also be interpreted as the reset of an integrate-and-fire neuron. Later, when we generalise to networks of neurons, we shall employ this interpretation.

[2]The read-out weights must scale as $\Gamma \sim 1/N$ so that firing rates are not unrealistically small in large networks. We can see this by calculating the average firing rate $\sum_{i=1}^{N} \bar{o}_i/N \approx x/(\Gamma N) \sim \mathcal{O}(N/N) \sim \mathcal{O}(1)$.

[3]The first neuron to spike will be random if there is some membrane potential noise.

[4]From the definition of the membrane potential we can see that $V_k \sim \mathcal{O}(1/N)$ because $\Gamma \sim 1/N$. Therefore, the size of the first term in Eqn. (18) is $\sum_k V_k \delta_{ki} \bar{o}_j = V_i \bar{o}_j \sim \mathcal{O}(1/N)$. Therefore, the second term can be ignored if $\sum_{kl} V_k \Omega_{kl} \partial \bar{o}_l / \partial \Omega_{ij} \ll \mathcal{O}(1/N)$. This happens if $\Omega_{kl} \ll \mathcal{O}(1/N^2)$ as at the start of learning. It also happens towards the end of learning if the terms $\{\Omega_{kl} \partial \bar{o}_l / \partial \Omega_{ij}\}$ are weakly correlated with zero mean, or if the membrane potentials $\{V_i\}$ are weakly correlated with zero mean.

[5]Despite being greedy, this spiking rule can generate firing rates that are practically identical to the optimal solutions: we checked this numerically in a large ensemble of networks with randomly chosen kernels.

# References

[1] Tolhurst D, Movshon J, and Dean A (1982) The statistical reliability of signals in single neurons in cat and monkey visual cortex. *Vision Res* **23**: 775–785.

[2] Shadlen MN, Newsome WT (1998) The variable discharge of cortical neurons: implications for connectivity, computation, and information coding. *J Neurosci* **18**(10): 3870–3896.

[3] Zohary E, Newsome WT (1994) Correlated neuronal discharge rate and its implication for psychophysical performance. *Nature* **370**: 140–143.

[4] Renart A, de la Rocha J, Bartho P, Hollender L, Parga N, Reyes A, & Harris, KD (2010) The asynchronous state in cortical circuits. *Science* **327**, 587–590.

[5] Ecker AS, Berens P, Keliris GA, Bethge M, Logothetis NK, Tolias AS (2010) Decorrelated neuronal firing in cortical microcircuits. *Science* **327**: 584–587.

[6] Okun M, Lampl I (2008) Instantaneous correlation of excitation and inhibition during ongoing and sensory-evoked activities. *Nat Neurosci* **11**, 535–537.

[7] Shu Y, Hasenstaub A, McCormick DA (2003) Turning on and off recurrent balanced cortical activity. *Nature* **423**, 288–293.

[8] Gentet LJ, Avermann M, Matyas F, Staiger JF, Petersen CCH (2010) Membrane potential dynamics of GABAergic neurons in the barrel cortex of behaving mice. *Neuron* **65**: 422–435.

[9] Caporale N, Dan Y (2008) Spike-timing-dependent plasticity: a Hebbian learning rule. *Annu Rev Neurosci* **31**: 25–46.

[10] Boerlin M, Deneve S (2011) Spike-based population coding and working memory. *PLoS Comput Biol* **7**, e1001080.

[11] Boerlin M, Machens CK, Deneve S (2012) Predictive coding of dynamic variables in balanced spiking networks. *under review*.

[12] Clopath C, Büsing L, Vasilaki E, Gerstner W (2010) Connectivity reflects coding: a model of voltage-based STDP with homeostasis. *Nat Neurosci* **13**(3): 344–352.

[13] van Vreeswijk C, Sompolinsky H (1998) Chaotic balanced state in a model of cortical circuits. *Neural Comput* **10**(6): 1321–1371.

[14] Brunel N (2000) Dynamics of sparsely connected networks of excitatory and inhibitory neurons. *J Comput Neurosci* **8**, 183–208.

[15] Vogels TP, Rajan K, Abbott LF (2005) Neural network dynamics. *Annu Rev Neurosci* **28**: 357–376.

[16] Vogels TP, Sprekeler H, Zenke F, Clopath C, Gerstner W. (2011) Inhibitory plasticity balances excitation and inhibition in sensory pathways and memory networks. *Science* **334**(6062):1569–73.

[17] Song S, Miller KD, Abbott LF (2000) Competitive Hebbian learning through spike-timing-dependent synaptic plasticity. *Nat Neurosci* **3**(9): 919–926.

